# Generalization in Reinforcement Learning: Safely Approximating the Value Function

**Justin A. Boyan** and **Andrew W. Moore**
Computer Science Department
Carnegie Mellon University
Pittsburgh, PA 15213
jab@cs.cmu.edu, awm@cs.cmu.edu

## Abstract

A straightforward approach to the curse of dimensionality in reinforcement learning and dynamic programming is to replace the lookup table with a generalizing function approximator such as a neural net. Although this has been successful in the domain of backgammon, there is no guarantee of convergence. In this paper, we show that the combination of dynamic programming and function approximation is not robust, and in even very benign cases, may produce an entirely wrong policy. We then introduce Grow-Support, a new algorithm which is safe from divergence yet can still reap the benefits of successful generalization.

## 1  INTRODUCTION

Reinforcement learning—the problem of getting an agent to learn to act from sparse, delayed rewards—has been advanced by techniques based on dynamic programming (DP). These algorithms compute a *value function* which gives, for each state, the minimum possible long-term cost commencing in that state. For the high-dimensional and continuous state spaces characteristic of real-world control tasks, a discrete representation of the value function is intractable; some form of generalization is required.

A natural way to incorporate generalization into DP is to use a function approximator, rather than a lookup table, to represent the value function. This approach, which dates back to uses of Legendre polynomials in DP [Bellman et al., 1963], has recently worked well on several dynamic control problems [Mahadevan and Connell, 1990, Lin, 1993] and succeeded spectacularly on the game of backgammon [Tesauro, 1992, Boyan, 1992]. On the other hand, many sensible implementations have been less successful [Bradtke, 1993, Schraudolph et al., 1994]. Indeed, given the well-established success

on backgammon, the absence of similarly impressive results appearing for other games is perhaps an indication that using function approximation in reinforcement learning does not always work well.

In this paper, we demonstrate that the straightforward substitution of function approximators for lookup tables in DP is not robust and, even in very benign cases, may diverge, resulting in an entirely wrong control policy. We then present Grow-Support, a new algorithm designed to converge robustly. Grow-Support grows a collection of states over which function approximation is stable. One-step backups based on Bellman error are not used; instead, values are assigned by performing "rollouts"—explicit simulations with a greedy policy. We discuss potential computational advantages of this method and demonstrate its success on some example problems for which the conventional DP algorithm fails.

## 2   DISCRETE AND SMOOTH VALUE ITERATION

Many popular reinforcement learning algorithms, including Q-learning and TD(0), are based on the dynamic programming algorithm known as value iteration [Watkins, 1989, Sutton, 1988, Barto *et al.*, 1989], which for clarity we will call *discrete value iteration*. Discrete value iteration takes as input a complete model of the world as a Markov Decision Task, and computes the optimal value function $J^*$:

$$J^*(x) = \text{the minimum possible sum of future costs starting from } x$$

To assure that $J^*$ is well-defined, we assume here that costs are nonnegative and that some absorbing goal state—with all future costs 0—is reachable from every state. For simplicity we also assume that state transitions are deterministic. Note that $J^*$ and the world model together specify a "greedy" policy which is optimal for the domain:

$$\text{optimal action from state } x = \arg\min_{a \in A} \left( \text{COST}(x, a) + J^*(\text{NEXT-STATE}(x, a)) \right)$$

We now consider extending discrete value iteration to the continuous case: we replace the lookup table over all states with a function approximator trained over a sample of states. The *smooth value iteration* algorithm is given in the appendix. Convergence is no longer guaranteed; we instead recognize four possible classes of behavior:

**good convergence** The function approximator accurately represents the intermediate value functions at each iteration (that is, after $m$ iterations, the value function correctly represents the cost of the cheapest $m$-step path), and successfully converges to the optimal $J^*$ value function.

**lucky convergence** The function approximator does not accurately represent the intermediate value functions at each iteration; nevertheless, the algorithm manages to converge to a value function whose greedy policy is optimal.

**bad convergence** The algorithm converges, i.e. the target J-values for the $N$ training points stop changing, but the resulting value function and policy are poor.

**divergence** Worst of all: small fitter errors may become magnified from one iteration to the next, resulting in a value function which never stops changing.

The hope is that the intermediate value functions will be smooth and we will achieve "good convergence." Unfortunately, our experiments have generated all four of these behaviors—and the divergent behavior occurs frequently, even for quite simple problems.

## 2.1   DIVERGENCE IN SMOOTH VALUE ITERATION

We have run simulations in a variety of domains—including a continuous gridworld, a car-on-the-hill problem with nonlinear dynamics, and tic-tac-toe versus a stochastic opponent—and using a variety of function approximators, including polynomial regression, backpropagation, and local weighted regression. In our experiments, none of these function approximators was immune from divergence.

The first set of results is from the **2-D continuous gridworld**, described in Figure 1. By quantizing the state space into a $100 \times 100$ grid, we can compute $J^*$ with discrete value iteration, as shown in Figure 2. The optimal value function is exactly linear: $J^*(x, y) = 20 - 10x - 10y$.

Since $J^*$ is linear, one would hope smooth value iteration could converge to it with a function approximator as simple as linear or quadratic regression. However, the intermediate value functions of Figure 2 are not smooth and cannot be fit accurately by a low-order polynomial. Using linear regression on a sample of 256 randomly-chosen states, smooth value iteration took over 500 iterations before "luckily" converging to optimal. Quadratic regression, though it always produces a smaller fit error than linear regression, did not converge (Figure 3). The quadratic function, in trying to both be flat in the middle of state space and bend down toward 0 at the goal corner, must compensate by underestimating the values at the corner opposite the goal. These underestimates then enlarge on each iteration, as the one-step DP lookaheads erroneously indicate that points can lower their expected cost-to-go by stepping farther away from the goal. The resulting policy is anti-optimal.

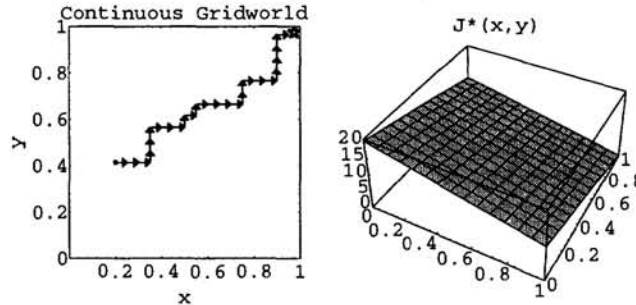

Figure 1: In the continuous gridworld domain, the state is a point $(x, y) \in [0, 1]^2$. There are four actions corresponding to short steps (length 0.05, cost 0.5) in each compass direction, and the goal region is the upper right-hand corner. $J^*(x, y)$ is linear.

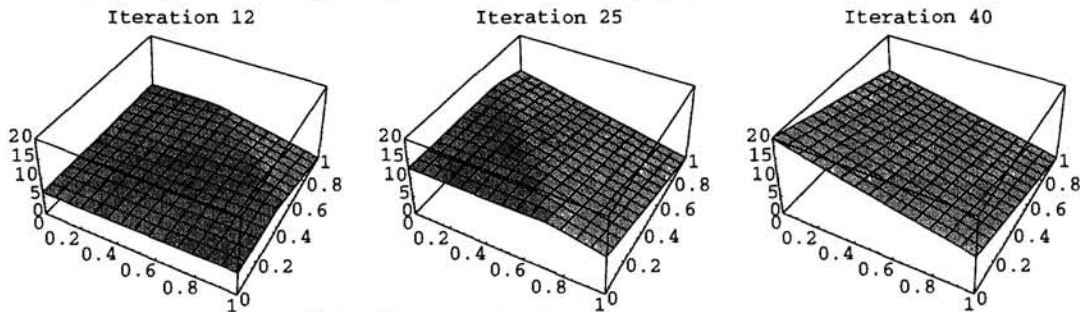

Figure 2: Computation of $J^*$ by discrete value iteration

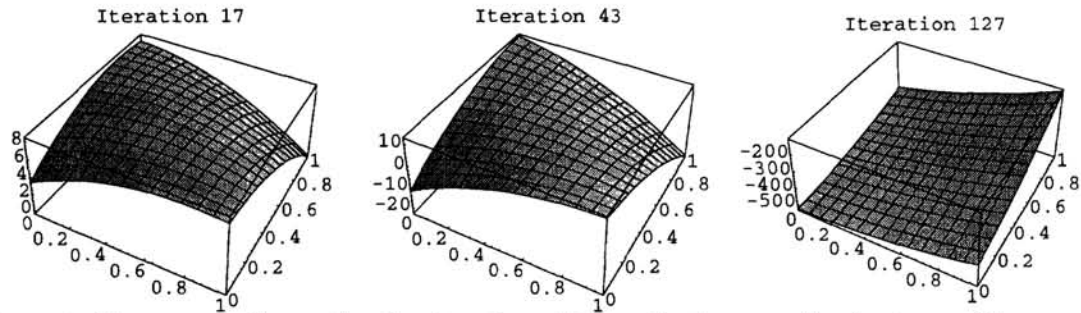

Figure 3: Divergence of smooth value iteration with quadratic regression (note z-axis).

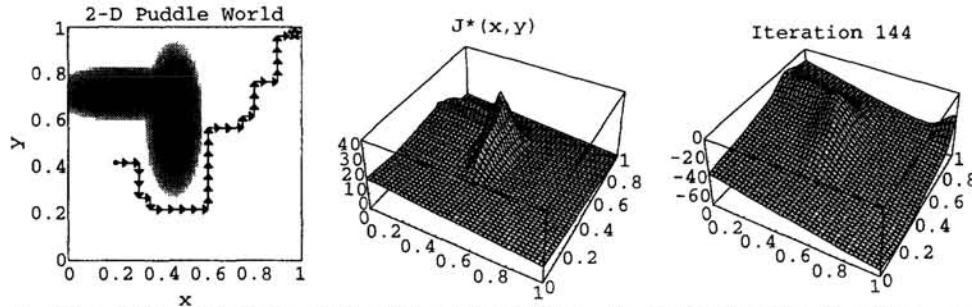

Figure 4: The 2-D continuous gridworld with puddles, its optimal value function, and a diverging approximation of the value function by Local Weighted Regression (note z-axis).

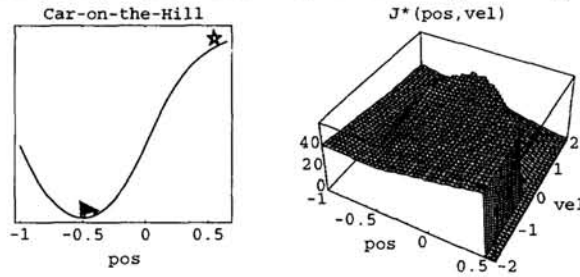

Figure 5: The car-on-the-hill domain. When the velocity is below a threshold, the car must reverse up the left hill to gain enough speed to reach the goal, so $J^*$ is discontinuous.

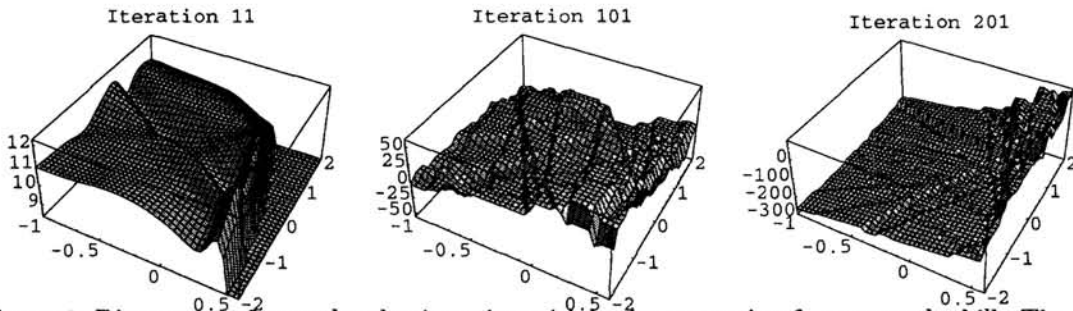

Figure 6: Divergence of smooth value iteration with backpropagation for car-on-the-hill. The neural net, a 2-layer MLP with 80 hidden units, was trained for 2000 epochs per iteration.

It may seem as though the divergence of smooth value iteration shown above can be attributed to the global nature of polynomial regression. In fact, when the domain is made slightly less trivial, the same types of instabilities appear with even a highly

Table 1: Summary of convergence results: Smooth value iteration

| Domain | Linear | Quadratic | LWR | Backprop |
|---|---|---|---|---|
| 2-D gridworld | lucky | diverge | good | lucky |
| 2-D puddle world | — | — | diverge | diverge |
| Car-on-the-hill | — | — | good | diverge |

local memory-based function approximator such as local weighted regression (LWR) [Cleveland and Delvin, 1988]. Figure 4 shows the continuous gridworld augmented to include two oval "puddles" through which it is costly to step. Although LWR can fit the corresponding $J^*$ function nearly perfectly, smooth value iteration with LWR nonetheless reliably diverges. On another two-dimensional domain, the car-on-the-hill (Figure 5), smooth value iteration with LWR did converge, but a neural net trained by backpropagation did not (see Figure 6). Table 1 summarizes our results.

In light of such experiments, we conclude that the straightforward combination of DP and function approximation is not robust. A general-purpose learning method will require either using a function approximator constrained to be robust during DP [Yee, 1992], or an algorithm which explicitly prevents divergence even in the face of imperfect function approximation, such as the Grow-Support algorithm we present in Section 3.

## 2.2 RELATED WORK
Theoretically, it is not surprising that inserting a smoothing process into a recursive DP procedure can lead to trouble. In [Thrun and Schwartz, 1993] one case is analyzed with the assumption that errors due to function approximation bias are independently distributed. Another area of theoretical analysis concerns inadequately approximated $J^*$ functions. In [Singh and Yee, 1994] and [Williams, 1993] bounds are derived for the maximum reduction in optimality that can be produced by a given error in function approximation. If a basis function approximator is used, then the reduction can be large [Sabes, 1993]. These results assume generalization from a dataset containing true optimal values; the true reinforcement learning scenario is even harder because each iteration of DP requires its own function approximation.

## 3 THE GROW-SUPPORT ALGORITHM
The Grow-Support algorithm is designed to construct the optimal value function with a generalizing function approximator while being robust and stable. It recognizes that function approximators cannot always be relied upon to fit the intermediate value functions produced by DP. Instead, it assumes only that the function approximator can represent the final $J^*$ function accurately. The specific principles of Grow-Support are these:

1. We maintain a "support" set of states whose final $J^*$ values have been computed, starting with goal states, and growing this set out from the goal. The fitter is trained only on these values, which we assume it is capable of fitting.

2. Instead of propagating values by one-step DP backups, we use simulations with the current greedy policy, called "rollouts". They explicitly verify the achievability of a state's cost-to-go estimate before adding that state to the

support. In a rollout, the $J$ values are derived from costs of actual paths to the goal, not from the values of the previous iteration's function approximation. This prevents divergence.

3. We take maximum advantage of generalization. Each iteration, we add to the support set any sample state which can, by executing a single action, reach a state that passes the rollout test. In a discrete environment, this would cause the support set to expand in one-step concentric "shells" back from the goal. But in our continuous case, the function approximator may be able to extrapolate correctly well beyond the support region—and when this happens, we can add many points to the support set at once. This leads to the very desirable behavior that the support set grows in big jumps in regions where the value function is smooth.

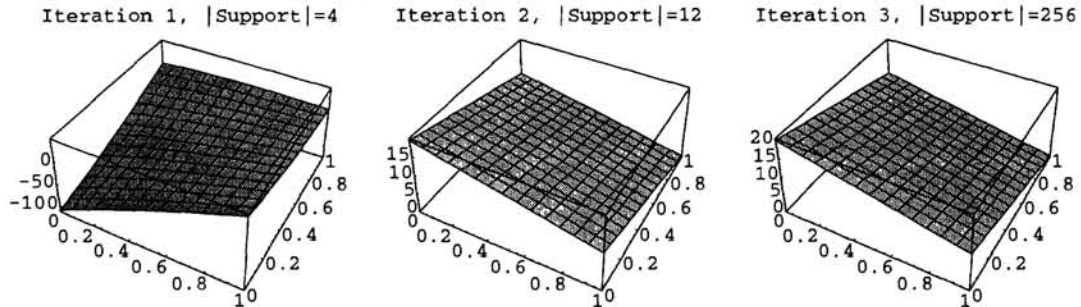

Figure 7: Grow-Support with quadratic regression on the gridworld. (Compare Figure 3.)

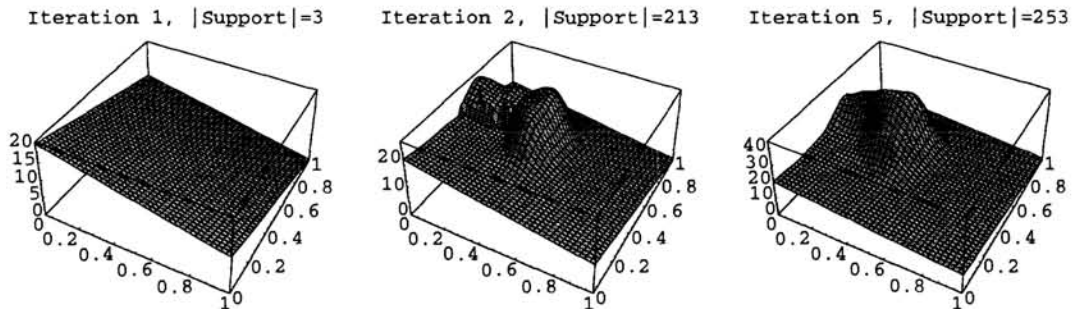

Figure 8: Grow-Support with LWR on the two-puddle gridworld. (Compare Figure 4.)

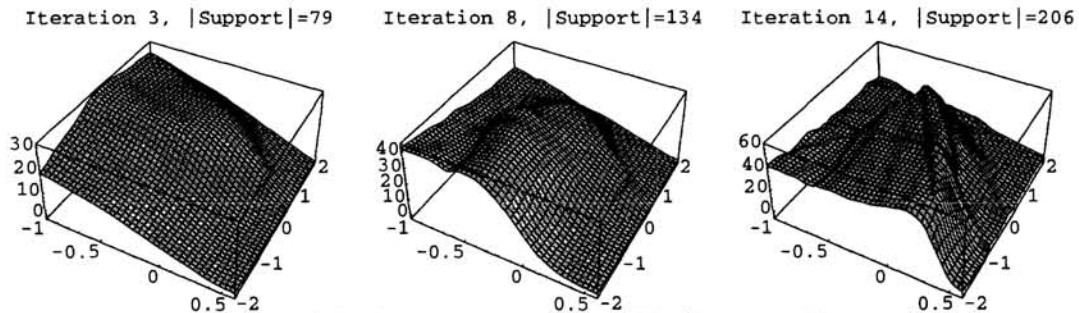

Figure 9: Grow-Support with backprop on car-on-the-hill. (Compare Figure 6.)

The algorithm, again restricted to the deterministic case for simplicity, is outlined in the appendix. In Figures 7–9, we illustrate its convergence on the same combinations of domain and function approximator which caused smooth value iteration to diverge. In Figure 8, all but three points are added to the support within only five iterations,

and the resulting greedy policy is optimal. In Figure 9, after 14 iterations, the algorithm terminates. Although 50 states near the discontinuity were not added to the support set, the resulting policy is optimal within the support set. Grow-support converged to a near-optimal policy for all the problems and fitters in Table 1.

The Grow-Support algorithm is more robust than value iteration. Empirically, it was also seen to be no more computationally expensive (and often much cheaper) despite the overhead of performing rollouts. Reasons for this are (1) the rollout test is not expensive; (2) once a state has been added to the support, its value is fixed and it needs no more computation; and most importantly, (3) the aggressive exploitation of generalization enables the algorithm to converge in very few iterations. However, with a nondeterministic problem, where multiple rollouts are required to assess the accuracy of a prediction, Grow-Support would become more expensive.

It is easy to prove that Grow-Support will always terminate after a finite number of iterations. If the function approximator is inadequate for representing the $J^*$ function, Grow-Support may terminate before adding all sample states to the support set. When this happens, we then know exactly which of the sample states are having trouble and which have been learned. This suggests potential schemes for adaptively adding sample states to the support in problematic regions. Investigation of these ideas is in progress.

In conclusion, we have demonstrated that dynamic programming methods may diverge when their tables are replaced by generalizing function approximators. Our Grow-Support algorithm uses rollouts, rather than one-step backups, to assign training values and to keep inaccurate states out of the training set. We believe these principles will contribute substantially to producing practical, robust, reinforcement learning.

### Acknowledgements

We thank Scott Fahlman, Geoff Gordon, Mary Lee, Michael Littman and Marc Ringuette for their suggestions, and the NDSEG fellowship and NSF Grant IRI-9214873 for their support.

# APPENDIX: ALGORITHMS

---

Smooth Value Iteration($X, G, A$, NEXT-STATE, COST, FITJ):

Given: • a finite collection of states $\hat{X} = \{x_1, x_2, \ldots x_N\}$ sampled from the continuous state space $X \subset \Re^n$, and goal region $G \subset X$
 • a finite set of allowable actions $A$
 • a deterministic transition function NEXT-STATE $: X \times A \rightarrow X$
 • the 1-step cost function COST $: X \times A \rightarrow \Re$
 • a smoothing function approximator FITJ

iter := 0
$j^{(0)}[i] := 0 \quad \forall i = 1 \ldots N$
repeat
  Train FITJ$^{(iter)}$ to approximate the training set: $\left\{ \begin{array}{c} x_1 \mapsto j^{(iter)}[1] \\ \vdots \\ x_N \mapsto j^{(iter)}[N] \end{array} \right\}$
  iter := iter + 1;
  for $i := 1 \ldots N$ do

$$j^{(iter)}[i] := \left\{ \begin{array}{ll} 0 & \text{if } x_i \in G \\ \min_{a \in A} \left( \text{COST}(x_i, a) + \text{FITJ}^{(iter-1)}(\text{NEXT-STATE}(x_i, a)) \right) & \text{otherwise} \end{array} \right.$$

until $j$ array stops changing

---

subroutine **RolloutCost**$(x, J)$:
    Starting from state $x$, follow the greedy policy defined by value function $J$ until
        either reaching the goal, or exceeding a total path cost of $J(x) + \epsilon$. Then return:
        $\longrightarrow$ the actual total cost of the path, if goal is reached from $x$ with cost $\leq J(x) + \epsilon$ ;
        $\longrightarrow \infty$, if goal is not reached in cost $J(x) + \epsilon$.

**Grow-Support**$(X, G, A, \text{NEXT-STATE}, \text{COST}, \text{FITJ})$:
    Given: • exactly the same inputs as Smooth Value Iteration.
SUPPORT $:= \{(x_i \mapsto 0) \mid x_i \in G\}$
repeat
    Train FITJ to approximate the training set SUPPORT
    for each $x_i \notin$ SUPPORT do
        $c := \min_{a \in A} [\text{COST}(x_i, a) + \textbf{RolloutCost}(\text{NEXT-STATE}(x_i, a), \text{FITJ})]$
        if $c < \infty$ then
            add $(x_i \mapsto c)$ to the training set SUPPORT
until SUPPORT stops growing or includes all sample points.

# References

[Barto *et al.*, 1989] A. Barto, R. Sutton, and C. Watkins. Learning and sequential decision making. Technical Report COINS 89-95, Univ. of Massachusetts, 1989.

[Bellman *et al.*, 1963] R. Bellman, R. Kalaba, and B. Kotkin. Polynomial approximation—a new computational technique in dynamic programming: Allocation processes. *Mathematics of Computation*, 17, 1963.

[Boyan, 1992] J. A. Boyan. Modular neural networks for learning context-dependent game strategies. Master's thesis, Cambridge University, 1992.

[Bradtke, 1993] S. J. Bradtke. Reinforcement learning applied to linear quadratic regulation. In S. J. Hanson, J. Cowan, and C. L. Giles, editors, *NIPS-5*. Morgan Kaufmann, 1993.

[Cleveland and Delvin, 1988] W. S. Cleveland and S. J. Delvin. Locally weighted regression: An approach to regression analysis by local fitting. *JASA*, 83(403):596–610, September 1988.

[Lin, 1993] L.-J. Lin. *Reinforcement Learning for Robots Using Neural Networks*. PhD thesis, Carnegie Mellon University, 1993.

[Mahadevan and Connell, 1990] S. Mahadevan and J. Connell. Automatic programming of behavior-based robots using reinforcement learning. Technical report, IBM T. J. Watson Research Center, NY 10598, 1990.

[Sabes, 1993] P. Sabes. Approximating Q-values with basis function representations. In *Proceedings of the Fourth Connectionist Models Summer School*, 1993.

[Schraudolph *et al.*, 1994] N. Schraudolph, P. Dayan, and T. Sejnowski. Using TD($\lambda$) to learn an evaluation function for the game of Go. In J. D. Cowan, G. Tesauro, and J. Alspector, editors, *NIPS-6*. Morgan Kaufmann, 1994.

[Singh and Yee, 1994] S. P. Singh and R. Yee. An upper bound on the loss from approximate optimal-value functions. *Machine Learning*, 1994. Technical Note (to appear).

[Sutton, 1988] R. Sutton. Learning to predict by the methods of temporal differences. *Machine Learning*, 3, 1988.

[Tesauro, 1992] G. Tesauro. Practical issues in temporal difference learning. *Machine Learning*, 8(3/4), May 1992.

[Thrun and Schwartz, 1993] S. Thrun and A. Schwartz. Issues in using function approximation for reinforcement learning. In *Proceedings of the Fourth Connectionist Models Summer School*, 1993.

[Watkins, 1989] C. Watkins. *Learning from Delayed Rewards*. PhD thesis, Cambridge University, 1989.

[Williams, 1993] R. Williams. Tight performance bounds on greedy policies based on imperfect value functions. Technical Report NU-CCS-93-13, Northeastern University, 1993.

[Yee, 1992] R. Yee. Abstraction in control learning. Technical Report COINS 92-16, Univ. of Massachusetts, 1992.
